# A computer modeling approach to understanding the inferior olive and its relationship to the cerebellar cortex in rats

**Maurice Lee and James M. Bower**
Computation and Neural Systems Program
California Institute of Technology
Pasadena, CA 91125

## ABSTRACT

This paper presents the results of a simulation of the spatial relationship between the inferior olivary nucleus and folium crus IIA of the lateral hemisphere of the rat cerebellum. The principal objective of this modeling effort was to resolve an apparent conflict between a proposed zonal organization of olivary projections to cerebellar cortex suggested by anatomical tract-tracing experiments (Brodal & Kawamura 1980; Campbell & Armstrong 1983) and a more patchy organization apparent with physiological mapping (Robertson 1987). The results suggest that several unique features of the olivocerebellar circuit may contribute to the appearance of zonal organization using anatomical techniques, but that the detailed patterns of patchy tactile projections seen with physiological techniques are a more accurate representation of the afferent organization of this region of cortex.

## 1 INTRODUCTION

Determining the detailed anatomical structure of the nervous system has been a major focus of neurobiology ever since anatomical techniques for looking at the fine structure of individual neurons were developed more than 100 years ago (Ramón y Cajal 1911). In more recent times, new techniques that allow labeling of the distant targets of groups of neurons have extended this investigation to include studies of the topographic relationships between different brain regions. In general, these so-called "tract-tracing" techniques have greatly extended our knowledge of the interrelationships between neural structures, often guiding and reinforcing the results of physiological investigations (DeYoe & Van Essen 1988). However, in some cases, anatomical and physiological techniques have been interpreted as producing conflicting results. One case, considered here, involves the pattern of neuronal projections from the *inferior olivary nucleus* to the

*cerebellar cortex.* In this paper we describe the results of a computer modeling effort, based on the structure of the olivocerebellar projection, intended to resolve this conflict.

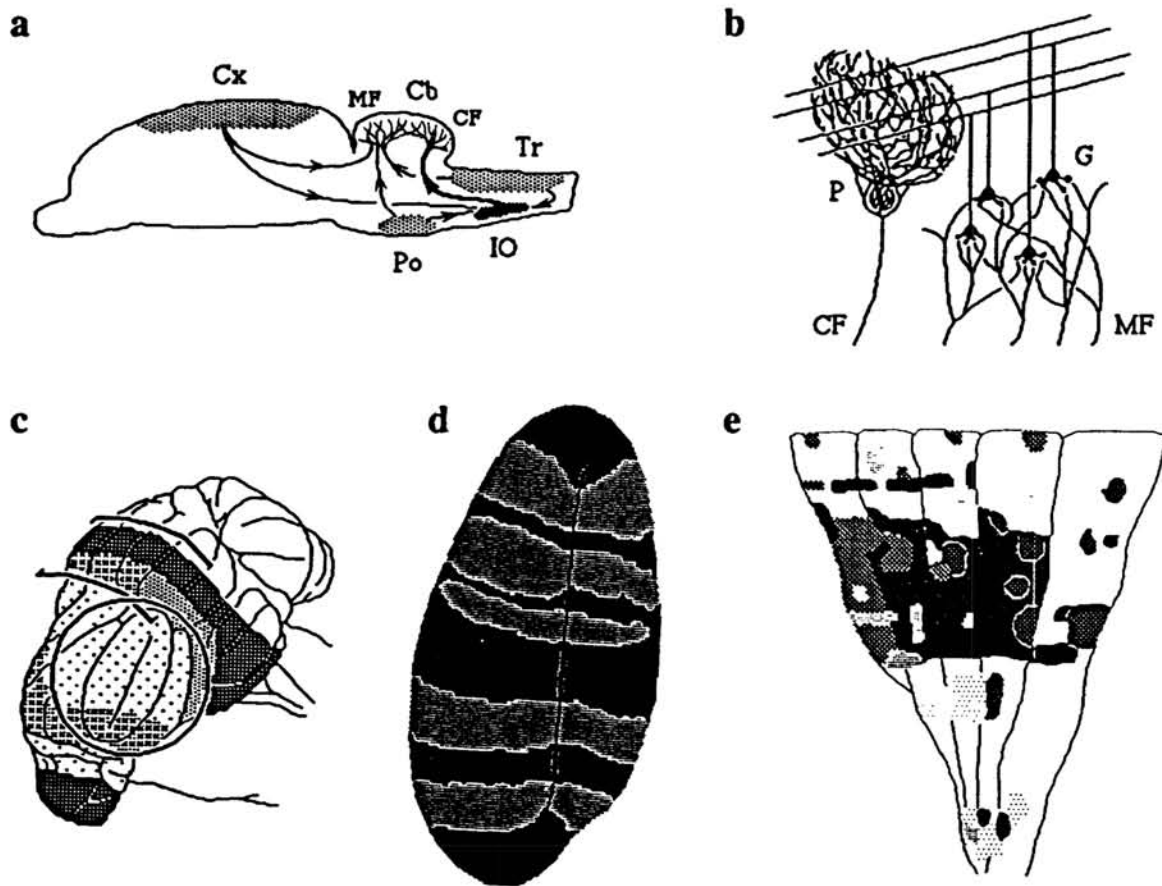

**Figure 1.** **a:** Profile of the rat brain, showing three areas (Cx, cerebral cortex; Po, pons; Tr, spinal trigeminal nucleus) that project to the cerebellum (Cb) via both climbing fiber (CF) pathways through the inferior olive (IO) and mossy fiber (MF) pathways. **b:** Magnified, highly simplified view of the cerebellar cortex, showing a Purkinje cell (P) being supplied with climbing fiber input, directly, and mossy fiber input, through the granule cells (G). **c:** Zonal organization of the olivocerebellar projection. *Different shading patterns* represent input from different areas of the inferior olive. Adapted from Campbell & Armstrong 1983. *Circled area* (crus IIA/crus IIB) is enlarged in Figure 1d; *bracketed area* (anterior lobe) is enlarged in Figure 1e. **d:** Detail of zonal organization. Dark areas represent bands of Purkinje cells that stain positive for monoclonal antibody Zebrin I. According to Gravel *et al.* 1987, these bands have boundaries similar to those resulting from partial tracer injections in the inferior olive. Adapted from Gundappa-Sulur *et al.* 1989. **e:** Patchy organization of the olivocerebellar projection (partial map). *Different shading patterns* represent input through the olive from different body surfaces. The horizontal and vertical scales are different. Adapted from Logan & Robertson 1986.

## 2  THE OLIVOCEREBELLAR SYSTEM

Purkinje cells, the principal neurons of the cerebellar cortex, are influenced by two major excitatory afferent projections to the cerebellum, the *mossy fiber system* and the *climbing fiber system* (Palay & Chan-Palay 1973). As shown in Figures 1a and 1b, mossy fibers arise from many different nuclei and influence Purkinje cells through granule cells within the cortex. Within the cortex the mossy fiber-granule cell-Purkinje cell circuit is characterized by enormous divergence (a single mossy fiber may influence several thousand Purkinje cells) and convergence (a single Purkinje cell may be influenced by several hundred thousand mossy fibers). In contrast, as also shown in Figures 1a and 1b, climbing fibers arise from a single source, the inferior olive, and exhibit severely limited divergence (10-15 Purkinje cells) and convergence (1 Purkinje cell).

Because the inferior olive is the sole source of the climbing fiber projection to the entire cerebellar cortex, and each Purkinje cell receives only one climbing fiber, the spatial organization of the olivocerebellar circuit has been the subject of a large research effort (Brodal & Kawamura 1980). Much of this effort has involved anatomical tract-tracing techniques in which injections of neuronally absorbed substances are traced from the inferior olive to the cerebellum or vice versa. Based on this work it has been proposed that the entire cerebellum is organized as a series of strips or zones, oriented in a parasagittal plane (Figures 1c, 1d: Campbell & Armstrong 1983; Gravel *et al.* 1987). This principle of organization has served as the basis for several functional speculations on the role of the cerebellum in coordinating movements (Ito 1984; Oscarsson 1980). Unfortunately, as suggested in the introduction, these anatomical results are somewhat at odds with the pattern of organization revealed by detailed electrophysiological mapping studies of olivary projections (Robertson 1987). Physiological results, summarized in Figure 1e, suggest that rather than being strictly zone-like, the olivocerebellar projection is organized more as a mosaic of parasagittally elongated patches.

## 3  THE MODEL

Our specific interests are with the tactilely responsive regions of the lateral hemispheres of the rat cerebellum (Bower *et al.* 1981; Welker 1987), and the modeling effort described here is a first step in using structural models to explore the functional organization of this region. As with previous modeling efforts in the olfactory system (Bower 1990), the current model is based on features of the anatomy and physiology of the real system. In the following section we will briefly describe these features.

### 3.1  ANATOMICAL ORGANIZATION

**Structure of the inferior olive.**  The inferior olive has a complex, highly folded conformation (Gwyn *et al.* 1977). The portion of the olive simulated in the model consists of a folded slab of 2520 olivary neurons with a volume of approximately 0.05 mm$^3$ (Figure 2a).

**Afferent projections to the olive.**  While inputs of various kinds and origins converge on this nucleus, we have limited those simulated here to tactile afferents from those

perioral regions known to influence the lateral cerebellar hemispheres (Shambes *et al.* 1978). These have been mapped to the olive following the somatotopically organized pattern suggested by several previous experiments (Gellman *et al.* 1983).

**Structure of the cerebellum.** The cerebellum is represented in the model by a flat sheet of 2520 Purkinje cells with an area of approximately 2 mm² (Figure 2a). Within this region, each Purkinje cell receives input from one, and only one, olivary neuron. Details of Purkinje cells at the cellular level have not been included in the current model.

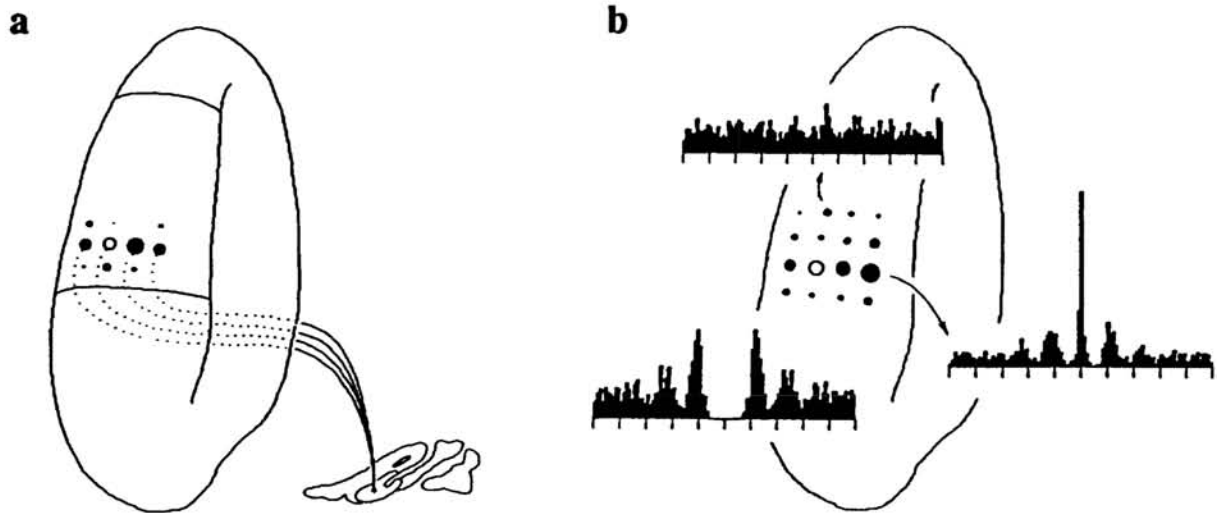

**Figure 2.** **a**: Basic structure of the model. Folia crus IIA and crus IIB of the cerebellum and a cross section of the inferior olive are shown, roughly to scale. The regions simulated in the model are outlined. *Clusters* of neighboring olivary neurons project to parasagittal *strips* of Purkinje cells as indicated. This figure also shows simulated correlation results similar to those in Figure 1b. **b**: Spatial structure of correlations among records of climbing fiber activity in crus IIA. Sizes of filled circles represent cross-correlation coefficients with respect to the "master" site (open circle). Sample cross-correlograms are shown for two sites as indicated. The autocorrelogram for the "master" site is also shown. Adapted from Sasaki *et al.* 1989.

## 3.2 PHYSIOLOGICAL ORGANIZATION

**Spatially correlated patterns of activity.** When the activities of multiple climbing fibers are recorded from within cerebellar cortex, there is a strong tendency for climbing fibers supplying Purkinje cells oriented parasagittally with respect to each other to be correlated in their firing activity (Sasaki *et al.* 1989: Figure 2b). It has been suggested that these correlations reflect the fact that direct electrotonic couplings exist between olivary neurons (Llinás & Yarom 1981a, b; Benardo & Foster 1986). These physiological results are simulated in two ways in the current model. First, neighboring olivary neurons are electrotonically coupled, thus firing in a correlated manner. Second, small clusters of olivary neurons have been made to project to parasagittally oriented strips of Purkinje

cells. Under these constraints, the model replicates the parasagittal pattern of climbing fiber activity found in certain regions of cerebellar cortex (compare Figures 2a and 2b).

**Topography of cerebellar afferents.** As discussed above, this model is intended to explore spatial and functional relationships between the inferior olive and the lateral hemispheres of the rat cerebellum. Unfortunately, a physiological map of the climbing fiber projections to this cerebellar region does not yet exist for the rat. However, a detailed map of mossy fiber tactile projections to this region is available (Welker 1987). As in the climbing fiber map in the anterior lobe (Robertson 1987; Figure 1e) and mossy fiber maps in various areas in the cat (Kassel *et al*. 1984), representations of different parts of the body surface are grouped into patches with adjacent patches receiving input from nonadjacent peripheral regions. On the assumption that the mossy fiber and climbing fiber maps coincide, we have based the modeled topography of the olivary projection to the cerebellum on the well-described mossy fiber map (Figure 3a). In the model, the smoothly varying topography of the olive is transformed to the patchy organization of the cerebellar cortex through the projection pathways taken to the cerebellum by different climbing fibers.

a                                                      b

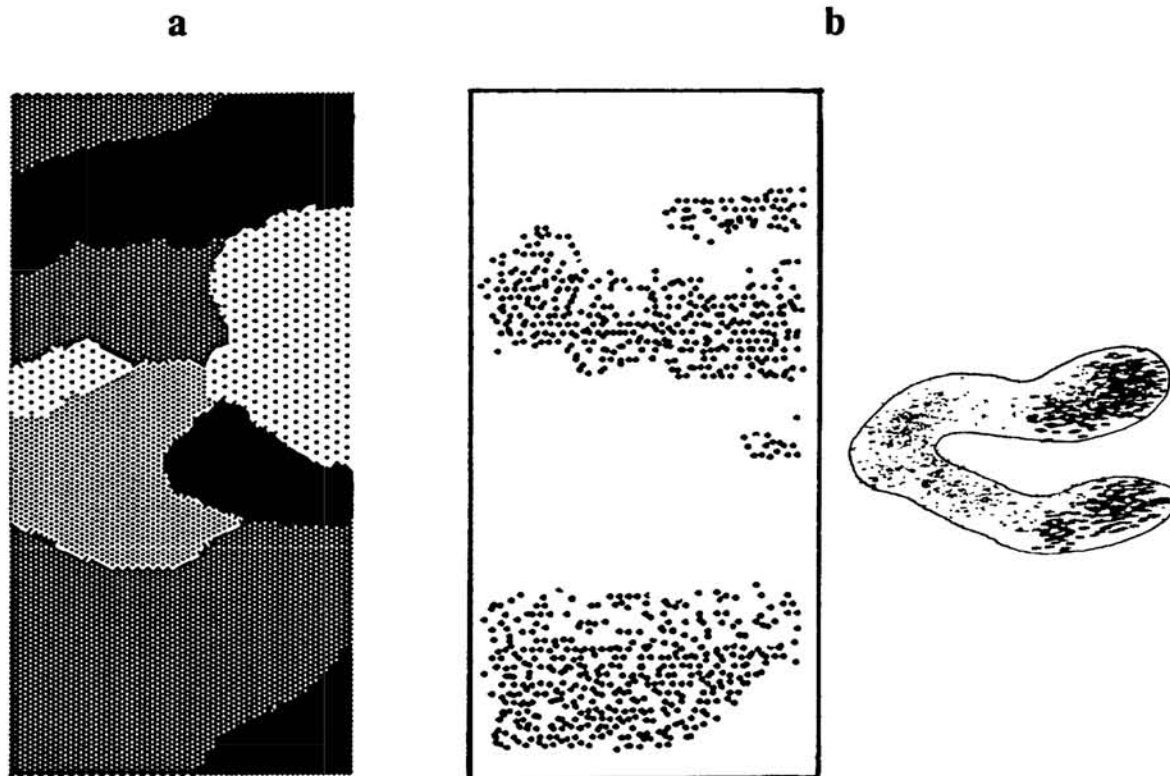

**Figure 3.** a: Organization of receptive field map in simulated region of crus IIA. *Different shading patterns* represent input from different perioral surfaces. b: Simulated tract-tracing experiment. *Left*, tracer visualization (dark areas) in the cerebellum. *Right*, tracer uptake (dark areas) in the inferior olive.

## 4 RESULTS: SIMULATION OF ZONAL ORGANIZATION

Having constructed the model to include each of the physiological features described above, we proceeded to replicate anatomical tract-tracing experiments. This was done by simulating the chemical labeling of neurons within restricted areas of inferior olive and following their connections to the cerebellum. As in the biological experiments, in many cases simulated injections included several folds of the olivary nucleus (Figure 3b). The results (Figure 3b) demonstrate patterns of labeling remarkably similar to those seen with real olivary injections in the rat (compare Figures 1d and 3b).

## 5 CONCLUSIONS AND FURTHER WORK

These simulation results have demonstrated that a broadly parasagittal organization can be generated in a model system which is actually based on a fine-grained patchy pattern of afferent projections. Further, the simulations allow us to propose that the appearance of parasagittal zonation may result from several unusual features of the olivary nucleus. First, the folding characteristic of the inferior olive likely places neurons with different receptive fields within a common area of tracer uptake in any given anatomical experiment, resulting in co-labeling of functionally different regions. Second, the tendency for local clusters of olivary neurons to project to parasagittal strips of Purkinje cells could serve to extend tracer injection in the parasagittal direction, enhancing the impression of parasagittal zones. This is further reinforced by the tendency of the patches themselves to be somewhat elongated in the parasagittal plane. Finally, the restricted resolution of the anatomical techniques could very well contribute to the overall impression of parasagittal zonation by obscuring small, unlabeled regions more apparent using physiological procedures. Modeling efforts currently under way will extend these results to more than one cerebellar folium in an attempt to account for the appearence of transfolial zones in some preparations.

In addition to these interpretations of previous data, this model also provides both directions for further physiological experiments and predictions concerning the results. First, the model assumes that mossy fiber and climbing fiber projections representing the same regions of the rat's body surface overlap in the cerebellum. We take the similarity in modeled and real tract-tracing results (Figures 1d and 3b) as suggesting strongly that this is, in fact, the case; however, physiological experiments are currently underway to test this hypothesis. Second, the model predicts that the parasagittal pattern of climbing fiber correlations found in a particular cerebellar region will be dependent on the pattern of tactile patches found in that region. Those regions containing large patches (*e.g.* the center of crus IIA) should clearly show parasagittal strips of correlated climbing fiber activity. However, in cortical regions containing smaller, more diverse sets of patches (*e.g.* more medial regions of crus IIA), this correlation structure should not be as clear. Experiments are also under way to test this prediction of the model.

## Acknowledgements

This model has been constructed using GENESIS, the Caltech neural simulation system. Simulation code for the model presented here can be accessed by registered GENESIS users. Information on the simulator or this model can be obtained from genesis@caltech.bitnet. This work was supported by NIH grant BNS 22205.

## References

Benardo, L. S., and R. E. Foster 1986. Oscillatory behavior in inferior olive neurons: Mechanism, modulation, cell aggregates. *Brain Res. Bull.* 17:773-784.

Bower, J. M. 1990. Reverse engineering the nervous system: An anatomical, physiological, and computer based approach. In *An introduction to neural and electronic networks*, ed. S. Zornetzer, J. Davis, and C. Lau, pp. 3-24. Academic Press.

Bower, J. M., and J. Kassel 1989. Variability in tactile projection patterns to crus IIA of the Norway rat. *J. Neurosci.* (submitted for publication).

Bower, J. M., D. H. Beermann, J. M. Gibson, G. M. Shambes, and W. Welker 1981. Principles of organization of a cerebro-cerebellar circuit. Micromapping the projections from cerebral (SI) to cerebellar (granule cell layer) tactile areas of rats. *Brain Behav. Evol.* 18:1-18.

Brodal, A., and K. Kawamura 1980. Olivocerebellar projection: A review. *Adv. Anat. Embryol. Cell Biol.* 64:1-140.

Campbell, N. C., and D. M. Armstrong 1983. Topographical localization in the olivocerebellar projection in the rat: An autoradiographic study. *Brain Res.* 275:235-249.

DeYoe, E. A., and D. C. Van Essen 1988. Concurrent processing streams in monkey visual cortex. *Trends Neurosci.* 11:219-226.

Gellman, R., J. C. Houk, and A. R. Gibson 1983. Somatosensory properties of the inferior olive of the cat. *J. Comp. Neurol.* 215:228-243.

Gravel, C., L. M. Eisenman, R. Sasseville, and R. Hawkes 1987. Parasagittal organization of the rat cerebellar cortex: Direct correlation between antigenic Purkinje cell bands revealed by mabQ113 and the organization of the olivocerebellar projection. *J. Comp. Neurol.* 265:294-310.

Gundappa-Sulur, G., H. Shojaeian, M. Paulin, L. Posakony, R. Hawkes, and J. M. Bower 1989. Variability in and comparisons of: 1) tactile projections to the granule cell layers of cerebellar cortex; and 2) the spatial distribution of Zebrin I-labeled Purkinje cells. *Soc. Neurosci. Abstr.* 15:612.

Gwyn, D. G., G. P. Nicholson, and B. A. Flumerfelt 1977. The inferior olivary nucleus of the rat: A light and electron microscopic study. *J. Comp. Neurol.* 174:489-520.

Ito, M. 1984. *The cerebellum and neural control.* Raven Press.

Kassel, J., G. M. Shambes, and W. Welker 1984. Fractured cutaneous projections to the granule cell layer of the posterior cerebellar hemispheres of the domestic cat. *J. Comp. Neurol.* 225:458-468.

Llinás, R., and Y. Yarom 1981a. Electrophysiology of mammalian inferior olivary neurones in vitro. Different types of voltage-dependent ionic conductances. *J.*

*Physiol. (Lond.)* 315:549-567.

Llinás, R., and Y. Yarom 1981b. Properties and distribution of ionic conductances generating electroresponsiveness of mammalian inferior olivary neurones in vitro. *J. Physiol. (Lond.)* 315:568-584.

Logan, K., and L. T. Robertson 1986. Somatosensory representation of the cerebellar climbing fiber system in the rat. *Brain Res.* 372:290-300.

Oscarsson, O. 1980. Functional organization of olivary projection to the cerebellar anterior lobe. In *The inferior olivary nucleus: Anatomy and physiology*, ed. J. Courville, C. de Montigny, and Y. Lammare, pp. 279-289. Raven Press.

Palay, S. L., and V. Chan-Palay 1973. *Cerebellar cortex: Cytology and organization.* Springer-Verlag.

Ramón y Cajal, S. 1911. *Histologie du système nerveux de l'homme et des vertèbres.* Maloine.

Robertson, L. T. 1987. Organization of climbing fiber representation in the anterior lobe. In *New concepts in cerebellar neurobiology*, ed. J. S. King, pp. 281-320. Alan R. Liss.

Sasaki, K., J. M. Bower, and R. Llinás 1989. Multiple Purkinje cell recording in rodent cerebellar cortex. *Eur. J. Neurosci.* (submitted for publication).

Shambes, G. M., J. M. Gibson, and W. Welker 1978. Fractured somatotopy in granule cell tactile areas of rat cerebellar hemispheres revealed by micromapping. *Brain Behav. Evol.* 15:94-140.

Welker, W. 1987. Spatial organization of somatosensory projections to granule cell cerebellar cortex: Functional and connectional implications of fractured somatotopy (summary of Wisconsin studies). In *New concepts in cerebellar neurobiology*, ed. J. S. King, pp. 239-280. Alan R. Liss.